# Rate Distortion Codes in Sensor Networks:
# A System-level Analysis

**Tatsuto Murayama and Peter Davis**
NTT Communication Science Laboratories
Nippon Telegraph and Telephone Corporation
"Keihanna Science City", Kyoto 619-0237, Japan
{murayama,davis}@cslab.kecl.ntt.co.jp

## Abstract

This paper provides a system-level analysis of a scalable distributed sensing model for networked sensors. In our system model, a data center acquires data from a bunch of L sensors which each independently encode their noisy observations of an original binary sequence, and transmit their encoded data sequences to the data center at a combined rate R, which is limited. Supposing that the sensors use independent LDGM rate distortion codes, we show that the system performance can be evaluated for any given finite R when the number of sensors L goes to infinity. The analysis shows how the optimal strategy for the distributed sensing problem changes at critical values of the data rate R or the noise level.

## 1   Introduction

Device and sensor networks are shaping many activities in our society. These networks are being deployed in a growing number of applications as diverse as agricultural management, industrial controls, crime watch, and military applications. Indeed, sensor networks can be considered as a promising technology with a wide range of potential future markets [1]. Still, for all the promise, it is often difficult to integrate the individual components of a sensor network in a smart way. Although we see many breakthroughs in component devices, advanced software, and power managements, system-level understanding of the emerging technology is still weak. It requires a shift in our notion of "what to look for". It requires a study of collective behavior and resulting trade-offs. This is the issue that we address in this article. We demonstrate the usefulness of adopting new approaches by considering the following scenario.

Consider that a data center is interested in the data sequence, $\{X(t)\}_{t=1}^{\infty}$, which cannot be observed directly. Therefore, the data center deploys a bunch of $L$ sensors which each independently encodes its noisy observation of the sequence, $\{Y_i(t)\}_{t=1}^{\infty}$, without sharing any information, i.e., the sensors are not permitted to communicate and decide what to send to the data center beforehand. The data center collects separate samples from all the $L$ sensors and uses them to recover the original sequence. However, since $\{X(t)\}_{t=1}^{\infty}$ is not the only pressing matter which the data center must consider, the combined data rate $R$ at which the sensors can communicate with it is strictly limited. A formulation of decentralized communication with estimation task, the "CEO problem", was first proposed by Berger

and Zhang [2], providing a new theoretical framework for large scale sensing systems. In this outstanding work, some interesting properties of such systems have been revealed. If the sensors were permitted to communicate on the basis of their pooled observations, then they would be able to smooth out their independent observation noises entirely as $L$ goes to infinity. Therefore, the data center can achieve an arbitrary fidelity $D(R)$, where $D(\cdot)$ denotes the distortion rate function of $\{X(t)\}$. In particular, the data center recovers almost complete information if $R$ exceeds the entropy rate of $\{X(t)\}$. However, if the sensors are not allowed to communicate with each other, there does not exist a finite value of $R$ for which even infinitely many sensors can make $D$ arbitrarily small [2].

In this paper, we introduce a new analytical model for a massive sensing system with a finite data rate $R$. More specifically, we assume that the sensors use LDGM codes for rate distortion coding, while the data center recovers the original sequence by using optimal "majority vote" estimation [3]. We consider the distributed sensing problem of deciding the optimal number of sensors $L$ given the combined data rate $R$. Our asymptotic analysis successfully provides the performance of the whole sensing system when $L$ goes to infinity, where the data rate for an individual sensor information vanishes. Here, we exploit statistical methods which have recently been developed in the field of disordered statistical systems, in particular, the spin glass theory. The paper is organized as follows. In Section 2, we introduce a system model for the sensor network. Section 3 summarizes the results of our approach, where the following section provides the outline of our analysis. Conclusions are given in the last section.

## 2 System Model

Let $P(x)$ be a probability distribution common to $\{X(t)\} \in \mathcal{X}$, and $W(y|x)$ be a stochastic matrix defined on $\mathcal{X} \times \mathcal{Y}$, with $\mathcal{Y}$ denotes the common alphabet of $\{Y_i(t)\}$, where $i = 1, \cdots, L$ and $t \geq 1$. In the general setup, we assume that the instantaneous joint probability distribution in the form

$$\Pr[x, y_1, \cdots, y_L] = P(x) \prod_{i=1}^{L} W(y_i|x)$$

for the temporally memoryless source $\{X(t)\}_{t=1}^{\infty}$. Here, the random variables $Y_i(t)$ are conditionally independent when $X(t)$ is given, and the conditional probabilities $W[y_i(t)|x(t)]$ are identical for all $i$ and $t$. In this paper, we impose the binary assumptions to the problem, i.e., the data sequence $\{X(t)\}$ and its noisy observations $\{Y_i(t)\}$ are all assumed to be binary sequences. Therefore, the stochastic matrix can be parameterized as

$$W(y|x) = \begin{cases} 1-p, & \text{if } y = x \\ p, & \text{otherwise} \end{cases},$$

where $p \in [0,1]$ represents the observation noise. Note also that the alphabets have been selected as $\mathcal{X} = \mathcal{Y}$. Furthermore, for simplicity, we also assume that $P(x) = 1/2$ always holds, implying that a purely random source is observed.

At the encoding stage, a sensor $i$ encodes a block $\boldsymbol{y}_i = [y_i(1), \cdots, y_i(n)]^T$ of length $n$ from the noisy observation $\{y_i(t)\}_{t=1}^{\infty}$, into a block $\boldsymbol{z}_i = [z_i(1), \cdots, z_i(m)]^T$ of length $m$ defined on $\mathcal{Z}$. Hereafter, we take the Boolean representation of the binary alphabet $\mathcal{X} = \{0, 1\}$, therefore $\mathcal{Y} = \mathcal{Z} = \{0, 1\}$ as well. Let $\hat{\boldsymbol{y}}_i$ be a reproduction sequence for the block, and we have a known integer $m < n$. Then, making use of a Boolean matrix $A_i$ of dimensionality $n \times m$, we are to find an $m$ bit codeword sequence $\boldsymbol{z}_i = [z_i(1), \cdots, z_i(m)]^T$ which satisfies

$$\hat{\boldsymbol{y}}_i = A_i \boldsymbol{z}_i \pmod{2}, \tag{1}$$

where the fidelity criterion

$$D = \frac{1}{n} d_{\mathrm{H}}(\boldsymbol{y}_i, \hat{\boldsymbol{y}}_i) \tag{2}$$

holds [4]. Here the Hamming distance $d_{\mathrm{H}}(\cdot, \cdot)$ is used for the distortion measure. Note that we have applied modulo-2 arithmetic for the additive operation in (1). Let $A_i$ be characterized by $K$ ones per row and $C$ per column. The finite, and usually small, numbers $K$ and $C$ define a particular LDGM code family. The data center then collects the $L$ codeword sequences, $\boldsymbol{z}_1, \cdots, \boldsymbol{z}_L$. Since all the $L$ codewords are of the same length $m$, the combined data rate will be $R = L \times m/n$. Therefore, in our scenario, the data center deploys exchangeable sensors with fixed quality reproductions, $\hat{\boldsymbol{y}}_1, \cdots, \hat{\boldsymbol{y}}_L$. Lastly, the $t$th symbol of the estimate, $\hat{\boldsymbol{x}} = [\hat{x}(1), \cdots, \hat{x}(n)]^T$, is to be calculated by majority vote [3],

$$\hat{x}(t) = \begin{cases} 0, & \text{if } \hat{y}_1(t) + \cdots + \hat{y}_L(t) \leq L/2 \\ 1, & \text{otherwise} \end{cases} . \tag{3}$$

Therefore, overall performance of the system can be measured by the expected bit error frequency for decisions by the majority vote (3), $P_{\mathrm{e}} = \Pr[x \neq \hat{x}]$.

In this paper, we consider two limit cases of decentralization levels; (1) The extreme situation of $L \to \infty$, and (2) the case of $L = R$. The former case means that the data rate for an individual sensor information vanishes, while the latter case results in the transmission without coding techniques. In general, it is difficult to determine which level is optimal for the estimation, i.e., which scenario results in the smaller value of $P_{\mathrm{e}}$. Indeed, by using the rate distortion codes, the data center could use as many sensors as possible for a given $R$. However, the quality of the individual reproduction would be less informative. The best choice seems to depend largely on $R$, as well as $p$.

## 3 Main Results

For simplicity, we consider the following two solvable cases; $K = 2$ for $C \geq K$ and the optimal case of $K \to \infty$. Let $p$ be a given observation noise level, and $R$ the finite real value of a given combined data rate. Letting $L \to \infty$, we find the expected bit error frequency to be

$$P_{\mathrm{e}}(p, R) = \int_{-\infty}^{-(1-2p)c_g \sqrt{R}} dr\, \mathrm{N}(0, 1) \tag{4}$$

with the constant value

$$c_g = \begin{cases} \frac{1}{\sqrt{2}} \left[ \frac{\sqrt{\alpha}}{2} + \frac{2\ln 2}{\sqrt{\alpha}} - \left( \frac{\sqrt{\alpha}}{2} - \frac{\sigma^2}{\sqrt{\alpha}} \right) \langle \tanh^2 x \rangle_{\pi(x)} \right] & (K = 2) \\ \sqrt{2\ln 2} & (K \to \infty) \end{cases} \tag{5}$$

where the rescaled variance $\sigma^2 = \alpha \langle \hat{x}^2 \rangle_{\hat{\pi}(\hat{x})}$ and the *first step RSB* enforcement

$$-\frac{1}{2} + \frac{2}{\alpha} \ln 2 + \left( \frac{1}{2} - \frac{\sigma^2}{\alpha} \right) \langle \tanh^2 x \, (1 + 2x \operatorname{csch} x \operatorname{sech} x) \rangle_{\pi(x)} = 0$$

holds. Here $\mathrm{N}(X, Y)$ denotes the normal distribution with the mean $X$ and the variance $Y$. The rescaled variance $\sigma^2$ and the scale invariant parameter $\alpha$ is determined numerically, where we use the following notations.

$$\langle \cdot \rangle_{\pi(x)} = \int_{-\infty}^{\infty} \frac{dx}{\sqrt{2\pi\sigma^2}} \exp\left[ -\frac{x^2}{2\sigma^2} \right] (\cdot) ,$$

$$\langle \cdot \rangle_{\hat{\pi}(\hat{x})} = \int_{-1}^{+1} \frac{d\hat{x}}{\sqrt{2\pi\sigma^2}} (1 - \hat{x}^2)^{-1} \exp\left[ -\frac{(\tanh^{-1} \hat{x})^2}{2\sigma^2} \right] (\cdot) .$$

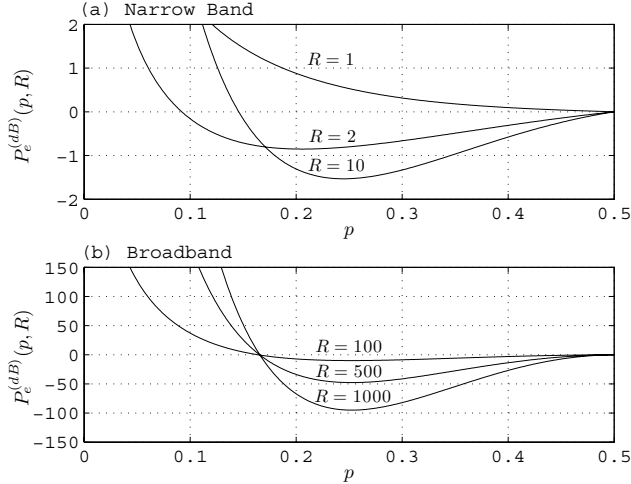

Figure 1: $P_{\mathrm{e}}^{(\mathrm{dB})}(p, R)$ for $K = 2$. (a) Narrow band (b) Broadband

Therefore, it is straightforward to evaluate (4) with (5) for given parameters, $p$ and $R$.

For a given *finite* value of $R$, we see what happens to the quality of the estimate when the noise level $p$ varies. Fig. 1 and Fig. 2 shows the typical behavior of the bit error frequency, $P_{\mathrm{e}}(p, R)$, in decibel (dB), where the reference level is chosen as

$$P_{\mathrm{e}}^{(0)}(p, R) = \begin{cases} \sum_{l=0}^{(R-1)/2} \binom{R}{l}(1-p)^l p^{R-l}, & (R \text{ is odd}) \\ \sum_{l=0}^{R/2-1} \binom{R}{l}(1-p)^l p^{R-l} + \frac{1}{2}\binom{R}{R/2}(1-p)^{R/2} p^{R/2} & (R \text{ is even}) \end{cases} \quad (6)$$

for a given integer $R$. The reference (6) denotes $P_{\mathrm{e}}$ for the case of $L = R$, i.e., the case when the sensors are not allowed to compress their observations. Here, in decibel, we have

$$P_{\mathrm{e}}^{(\mathrm{dB})}(p, R) = 10 \log \frac{P_{\mathrm{e}}(p, R)}{P_{\mathrm{e}}^{(0)}(p, R)} ,$$

where the $\log$ is to base 10. Note that the zero level in decibel occurs when the measured error frequency $P_{\mathrm{e}}(p, R)$ is equal to the reference level. Therefore, it is also possible to have negative levels, which would mean an expected bit error frequency much smaller than the reference level. In the case of small combined data rate $R$, the narrow band case, the numerical results in Fig. 1 (a) and Fig. 2 (a) show that the quality of the estimate is sensitive to the parity of the integer $R$. In particular, the $R = 2$ case has the lowest threshold level, $p_c = 0.0921$ for Fig. 1 (a) and $p_c = 0.082$ for Fig. 2 (a) respectively, beyond which the $L \to \infty$ scenario outperforms the $L = R$ scenario, while the $R = 1$ case does not have such a threshold. In contrast, if the bandwidth is wide enough, the difference of the expected bit error probabilities in decibel, $P_{\mathrm{e}}^{(\mathrm{dB})}(p, R)$, is proved to have similar qualitative characteristics as shown in Fig. 1 (b) and Fig. 2 (b). Moreover, our preliminary experiments for larger systems also indicate that the threshold $p_c$ seems to converge to the value, $0.165$ and $0.146$ respectively, as $L$ goes to infinity; we are currently working on the theoretical derivation.

## 4 Outline of Derivation

Since the predetermined matrices $A_1, \cdots, A_L$ are selected randomly, it is quite natural to say that the instantaneous series, defined by $\hat{\boldsymbol{y}}(t) = [\hat{y}_1(t), \cdots, \hat{y}_L(t)]^T$, can be modeled

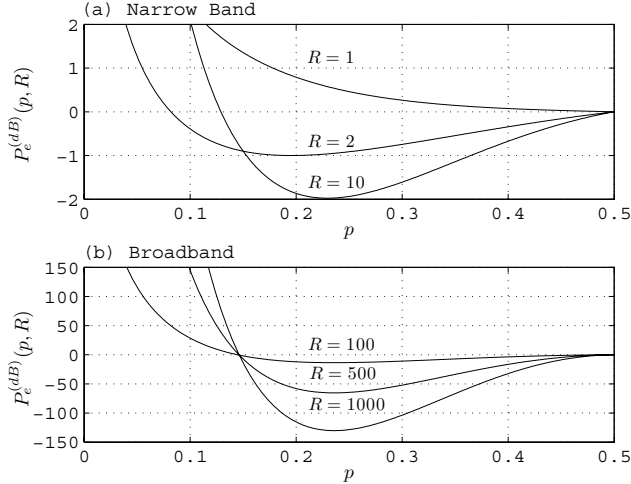

Figure 2: $P_{\mathrm{e}}^{(\mathrm{dB})}(p, R)$ for $K \to \infty$. (a) Narrow band (b) Broadband

using the Bernoulli trials. Here, the reproduction problem reduces to a channel model, where the stochastic matrix is defined as

$$W(\hat{y}|x) = \begin{cases} q, & \text{if } \hat{y} = x \\ 1 - q, & \text{otherwise} \end{cases}, \tag{7}$$

where $q$ denotes the quality of the reproductions, i.e., $\Pr[x \neq \hat{y}_i] = 1 - q$ for $i = 1, \cdots, L$. Letting the channel model (7) for the reproduction problem be valid, the expected bit error frequency can be well captured by using the cumulative probability distributions

$$P_{\mathrm{e}} = \Pr[x \neq \hat{x}] = \begin{cases} B(\frac{L-1}{2} : L, q), & \text{if } L \text{ is odd} \\ B(\frac{L}{2} - 1 : L, q) + \frac{1}{2} b(\frac{L}{2} : L, q) & \text{otherwise} \end{cases} \tag{8}$$

with

$$B(L' : L, q) = \sum_{l=0}^{L'} b(l : L, q), \quad b(l : L, q) = \binom{L}{l} q^l (1 - q)^{L-l},$$

where an integer $l$ be the total number of non-flipped elements in $\hat{\boldsymbol{y}}(t)$, and the second term $(1/2)b(L/2 : L, q)$ represents random guessing with $l = L/2$. Note that the reproduction quality $q$ can be easily obtained by the simple algebra $q = pD + (1 - p)(1 - D)$, where $D$ is the distortion with respect to coding.

Since the error probability (8) is given by a function of $q$, we firstly derive an analytical solution for the quality $q$ in the limit $L \to \infty$, keeping $R$ finite. In this approach, we apply the method of statistical mechanics to evaluate the *typical* performance of the codes [4]. As a first step, we translate the Boolean alphabets $\mathcal{Z} = \{0, 1\}$ to the "Ising" ones, $\mathcal{S} = \{+1, -1\}$. Consequently, we need to translate the additive operations, such as, $z_i(s) + z_i(s') \pmod 2$ into their multiplicative representations, $\sigma_i(s) \times \sigma_i(s') \in \mathcal{S}$ for $s, s' = 1, \cdots, m$. Similarly, we translate the Boolean $y_i(t)$s into the Ising $J_i(t)$s. For simplicity, we omit the subscript $i$, which labels the $L$ agents, in the rest of this section. Following the prescription of Sourlas [5], we examine the *Gibbs-Boltzmann distribution*

$$\Pr[\boldsymbol{\sigma}] = \frac{\exp\left[-\beta H(\boldsymbol{\sigma}|\boldsymbol{J})\right]}{Z(\boldsymbol{J})} \quad \text{with} \quad Z(\boldsymbol{J}) = \sum_{\boldsymbol{\sigma}} e^{-\beta H(\boldsymbol{\sigma}|\boldsymbol{J})}, \tag{9}$$

where the *Hamiltonian* of the Ising system is defined as

$$H(\boldsymbol{\sigma}|\boldsymbol{J}) = -\sum_{s_1<\cdots<s_K} \mathcal{A}_{s_1\ldots s_K} J_i[t(s_1,\ldots,s_K)]\sigma(s_1)\ldots\sigma(s_K)\,. \tag{10}$$

The observation index $t(s_1,\ldots,s_K)$ specifies the proper value of $t$ given the set $s_1,\ldots,s_K$, so that it corresponds to the parity check equation (1). Here the elements of the symmetric tensor $\mathcal{A}_{s_1\ldots s_K}$, representing dilution, is either zero or one depending on the set of indices $(s_1,\ldots,s_K)$. Since there are $C$ non-zero elements randomly chosen for any given index $s$, we find $\sum_{s_2,\ldots,s_K}\mathcal{A}_{ss_2\ldots s_K} = C$ . The code rate is $R/L = K/C$ because a reproduction sequence has $C$ bits per index $s$ and carries $K$ bits of the code-word. It is easy to see that the Hamiltonian (10) is counting the reproduction errors, $[1 - J_{t(s_1,\ldots,s_K)}\cdot\sigma(s_1)\ldots\sigma(s_K)]/2$.

Moreover, according to the statistical mechanics, we can easily derive the "observable" quantities using the *free energy* defined as

$$f = -\frac{1}{\beta}\left\langle \ln Z(\boldsymbol{J})\right\rangle_{\mathcal{A},\boldsymbol{J}}$$

which carries all information about the statistics of the system. Here, $\beta$ denotes an "inverse temperature" for the Gibbs-Boltzmann distribution (9), and $\langle\cdot\rangle_{\mathcal{A},\boldsymbol{J}}$ represents the configurational average. Therefore, we have to average the logarithm of the partition function $Z(\boldsymbol{J})$ over the given distribution $\langle\cdot\rangle_{\mathcal{A},\boldsymbol{J}}$ after the calculation of the partition function. Finally, to perform such a program, the *replica trick* is used [6]. The theory of *replica symmetry breaking* can provide the free energy resulting in the expression

$$\begin{aligned}
f = -\frac{1}{\beta n}\Bigg[ &\ln\cosh\beta - K\left\langle\ln\left[1 + \tanh(\beta x)\tanh(\beta\hat{x})\right]\right\rangle_{\pi(x),\hat{\pi}(\hat{x})} \\
&+ \frac{1}{2}\left\langle\sum_{J=\pm1}\ln\left[1 + \tanh(\beta J)\prod_{l=1}^{K}\tanh(\beta x_l)\right]\right\rangle_{\pi(\boldsymbol{x})} \\
&+ \frac{C}{K}\left\langle\ln\sum_{\sigma=\pm1}\prod_{l=1}^{C}[1 + \sigma\tanh(\beta\hat{x}_l)]\right\rangle_{\hat{\pi}(\hat{x})}\Bigg]\,,
\end{aligned} \tag{11}$$

where $\langle\cdot\rangle_{\pi(\boldsymbol{x})}$ denotes the averaging over $p(x_l)$s and so on. The variation of (11) by $\pi(\boldsymbol{x})$ and $\hat{\pi}(\hat{\boldsymbol{x}})$ under the condition of normalization gives the saddle point condition

$$\pi(x) = \left\langle\delta\left[x - \sum_{l=1}^{C-1}\hat{x}_l\right]\right\rangle_{\hat{\pi}(\hat{\boldsymbol{x}})}, \quad \hat{\pi}(\hat{x}) = \left\langle\frac{1}{2}\sum_{J=\pm1}\delta\left[\hat{x} - \mu(x_1,\ldots,x_{K-1};J)\right]\right\rangle_{\pi(\boldsymbol{x})},$$

where

$$\mu(x_1,\ldots,x_{K-1};J) = \frac{1}{\beta}\tanh^{-1}\left[\tanh(\beta J)\prod_{l=1}^{K-1}\tanh(\beta x_l)\right]\,.$$

We now investigate the case of $K = 2$. Applying the central limit theorem to $\pi(x)$ [7], we get

$$\pi(x) = \frac{1}{\sqrt{2\pi C\sigma^2}}e^{-\frac{x^2}{2C\sigma^2}}\,, \tag{12}$$

where $\sigma^2$ is the variance of $\hat{\pi}(\hat{x})$. Here the resulting distribution (12) is a even function. The leading contribution to $\mu$ is then given by $\mu(x;J) \sim J\cdot\tanh(\beta x)$ as $\beta$ goes to zero;

The expression is valid in the asymptotic region $L \gg 1$ for a fixed $R$. Then, the formula for the delta function yields [8]

$$
\begin{aligned}
\hat{\pi}(\hat{x}) &= \left\langle \delta\left[x - \frac{1}{\beta}\tanh^{-1}\hat{x}\right] \left|\rho'\left(\frac{1}{\beta}\tanh^{-1}\hat{x}; \hat{x}\right)\right|^{-1}\right\rangle_{\pi(x)} \\
&= \frac{(1-\hat{x}^2)^{-1}}{\sqrt{2\pi\beta^2 C\sigma^2}}\exp\left[-\frac{(\tanh^{-1}\hat{x})^2}{2\beta^2 C\sigma^2}\right],
\end{aligned}
\tag{13}
$$

where we have used $\rho(x; \hat{x}) = \hat{x} - \tanh(\beta x)$. Therefore, we have

$$
\sigma^2 = \langle\hat{x}^2\rangle_{\hat{\pi}(\hat{x})} = \int_{-1}^{+1}\frac{d\hat{x}}{\sqrt{2\pi\beta^2 C\sigma^2}}\frac{\hat{x}^2}{1-\hat{x}^2}\exp\left[-\frac{(\tanh^{-1}\hat{x})^2}{2\beta^2 C\sigma^2}\right]
$$

for given $\beta^2 C$. Inserting (12), (13) into (11), we get

$$
f = -\frac{\beta}{2} - \frac{R}{\beta}\ln 2 + \frac{1-2\sigma^2}{2}\beta\langle\tanh^2\tilde{x}\rangle_{\tilde{\pi}(\tilde{x})} \quad\text{with}\quad \tilde{\pi}(\tilde{x}) = \frac{1}{\sqrt{2\pi\beta^2 C\sigma^2}}e^{-\frac{\tilde{x}^2}{2\beta^2 C\sigma^2}},
$$

where we rewrite $\tilde{x} = \beta x$. The theory of *replica symmetry breaking* tells us that relevant value of $\beta$ should not be smaller than the "freezing point" $\beta_g$, which implies the vanishing entropy condition:

$$
\frac{\partial f}{\partial \beta} = -\frac{1}{2} + \frac{2}{\beta_g^2 C}\ln 2 + \frac{1-2\sigma^2}{2}\langle\tanh^2\tilde{x}\,(1 + 2\tilde{x}\,\mathrm{csch}\,\tilde{x}\,\mathrm{sech}\,\tilde{x})\rangle_{\tilde{\pi}(\tilde{x})} = 0.
$$

Accordingly, it is convenient for us to define a scaling invariant parameter $\alpha = \beta_g^2 C$, and to rewrite the variance $\tilde{\sigma}^2 = \alpha\sigma^2$ for simplicity. Introducing these newly defined parameters, the above results could be summarized as follows. Given $R$ and $L$, we find

$$
f = \sqrt{\frac{R}{L}}\left[-\frac{1}{2}\sqrt{\frac{\alpha}{2}} - \ln 2\sqrt{\frac{2}{\alpha}} + \sqrt{\frac{\alpha}{2}}\left(\frac{1}{2} - \frac{\tilde{\sigma}^2}{\alpha}\right)\langle\tanh^2\tilde{x}\rangle_{\tilde{\pi}(\tilde{x})}\right]
$$

with $\tilde{\sigma}^2 = \alpha\langle\hat{x}^2\rangle_{\hat{\pi}(\hat{x})}$, where the condition

$$
-\frac{1}{2} + \frac{2}{\alpha}\ln 2 + \left(\frac{1}{2} - \frac{\tilde{\sigma}^2}{\alpha}\right)\langle\tanh^2\tilde{x}\,(1 + 2\tilde{x}\,\mathrm{csch}\,\tilde{x}\,\mathrm{sech}\,\tilde{x})\rangle_{\tilde{\pi}(\tilde{x})} = 0 \tag{14}
$$

holds. Here we denote

$$
\langle\,\cdot\,\rangle_{\tilde{\pi}(\tilde{x})} = \int_{-\infty}^{\infty}\frac{d\tilde{x}}{\sqrt{2\pi\tilde{\sigma}^2}}\exp\left[-\frac{\tilde{x}^2}{2\tilde{\sigma}^2}\right](\,\cdot\,),
$$

$$
\langle\,\cdot\,\rangle_{\hat{\pi}(\hat{x})} = \int_{-1}^{+1}\frac{d\hat{x}}{\sqrt{2\pi\tilde{\sigma}^2}}(1-\hat{x}^2)^{-1}\exp\left[-\frac{(\tanh^{-1}\hat{x})^2}{2\tilde{\sigma}^2}\right](\,\cdot\,).
$$

Lastly, by using the cumulative probability distribution, we get

$$
P_{\mathrm{e}} = \sum_{l=0}^{L/2}\binom{L}{l}q^l(1-q)^{L-l} \sim \int_0^{L/2}dr\,\mathrm{N}(Lq, Lq(1-q)). \tag{15}
$$

It is easy to see that (15) can be converted to a standard normal distribution by changing variables to $\tilde{r} = (r - Lq)/\sqrt{Lq(1-q)}$ [7], so $d\tilde{r} = dr/\sqrt{Lq(1-q)}$, yielding

$$
P_{\mathrm{e}} \sim \int_{-\sqrt{L}}^{\tilde{r}_g}d\tilde{r}\,\mathrm{N}(0, 1)
$$

with

$$\tilde{r}_g = 2\sqrt{L}(1 - 2p)\left(D - \frac{1}{2}\right)$$

$$= \sqrt{\frac{R}{2}}(1 - 2p)\left[-\frac{1}{2}\sqrt{\alpha} - \frac{2\ln 2}{\sqrt{\alpha}} + \sqrt{\alpha}\left(\frac{1}{2} - \frac{\tilde{\sigma}^2}{\alpha}\right)\langle\tanh^2 \tilde{x}\rangle_{\tilde{\pi}(\tilde{x})}\right].$$

Note that the relation $D = (1 + f)/2$ holds at the vanishing entropy condition (14) [4]. Finally, we obtain the main result (4) in Section 3 in the limit $L \to \infty$, when we use proper notations for the variables and the name of the function.

We can investigate the asymptotic case of $K \to \infty$ in a similar way. Since the leading contribution to $\hat{\pi}(\hat{x})$ comes from the value of $x$ in the vicinity of $\sqrt{C\sigma^2}$, we find the expression $\hat{\pi}(\hat{x}) \approx \left\langle\delta\left[\hat{x} - y\beta^K(C\sigma^2)^{\frac{K}{2}}\right]\right\rangle$ by using the power counting. Therefore, within the Parisi RSB scheme, one obtain a set of equations

$$\sqrt{L}f = -\frac{\sqrt{\alpha_c}}{2} - \frac{R}{\sqrt{\alpha_c}}\ln 2 , \quad -\frac{1}{2} + \frac{R}{\alpha_c}\ln 2 = 0$$

with the scale-invariant $\alpha_c = \beta^2 L$. This results in $c_g = \sqrt{2\ln 2}$, as is mentioned before.

## 5  Conclusion

This paper provides a system-level perspective for massive sensor networks. The decentralized sensing problem argued in this paper was first addressed by Berger and his collaborators. However, this paper is the first work that gives a scheme to analyze practically tractable codes in the given finite data rate, and shows the existence of threshold level of noise of which the optimal levels of decentralization changes. Future work includes the theoretical derivation of the threshold level $p_c$ where $R$ goes to infinity, as well as the implementation problem.

### Acknowledgments

The authors thank Jun Muramatsu and Naonori Ueda for useful discussions. This work was supported by the Ministry of Education, Science, Sports and Culture (MEXT) of Japan, under the Grant-in-Aid for Young Scientists (B), 15760288.

## References

[1] (2005) Intel@Mote. [Online]. Available: http://www.intel.com/research/exploratory/motes.htm

[2] T. Berger, Z. Zhang, and H. Viswanathan, "The CEO problem," *IEEE Trans. Inform. Theory*, vol. 42, pp. 887–902, May 1996.

[3] D. J. C. MacKay, *Information Theory, Inference and Learning Algorithms*.  Cambridge, UK: Cambridge University Press, 2003.

[4] T. Murayama and M. Okada, "Rate distortion function in the spin glass state: a toy model," in *Advances in Neural Information Processing Systems 15 (NIPS'02)*, Denver, USA, Dec. 2002, pp. 423–430.

[5] N. Sourlas, "Spin-glass models as error-correcting codes," *Nature*, vol. 339, pp. 693–695, June 1989.

[6] V. Dotsenko, *Introduction to the Replica Theory of Disordered Statistical Systems*.  Cambridge, UK: Cambridge University Press, 2001.

[7] W. Hays, *Statistics (5th Edition)*.  Belmont, CA: Wadsworth Publishing, 1994.

[8] C. W. Wong, *Introduction to Mathematical Physics: Methods and Concepts*.  Oxford, UK: Oxford University Press, 1991.
